# A Rational Decision-Making Framework for Inhibitory Control

**Pradeep Shenoy**
Department of Cognitive Science
University of California, San Diego
pshenoy@ucsd.edu

**Rajesh P. N. Rao**
Department of Computer Science
University of Washington
rao@cs.washington.edu

**Angela J. Yu**
Department of Cognitive Science
University of California, San Diego
ajyu@ucsd.edu

## Abstract

Intelligent agents are often faced with the need to choose actions with uncertain consequences, and to modify those actions according to ongoing sensory processing and changing task demands. The requisite ability to dynamically modify or cancel planned actions is known as inhibitory control in psychology. We formalize inhibitory control as a rational decision-making problem, and apply to it to the classical stop-signal task. Using Bayesian inference and stochastic control tools, we show that the optimal policy systematically depends on various parameters of the problem, such as the relative costs of different action choices, the noise level of sensory inputs, and the dynamics of changing environmental demands. Our normative model accounts for a range of behavioral data in humans and animals in the stop-signal task, suggesting that the brain implements statistically optimal, dynamically adaptive, and reward-sensitive decision-making in the context of inhibitory control problems.

## 1   Introduction

In natural behavior as well as in engineering applications, there is often the need to choose, under time pressure, an action among multiple options with imprecisely known consequences. For example, consider the decision of buying a house. A wise buyer should collect sufficient data to make an informed decision, but waiting too long might mean missing out on a dream home. Thus, balanced against the informational gain afforded by lengthier deliberation is the opportunity cost of inaction. Further complicating matters is the possible occurrence of a rare and unpredictably timed adverse event, such as job loss or serious illness, that would require a dynamic reformulation of one's plan of action. This ability to dynamically modify or cancel a planned action that is no longer advantageous or appropriate is known as *inhibitory control* in psychology.

In psychology and neuroscience, inhibitory control has been studied extensively using the *stop-signal* (or countermanding) task [17]. In this task, subjects perform a simple two-alternative forced choice (2AFC) discrimination task on a *go* stimulus, whereby one of two responses is required depending on the stimulus. On a small fraction of trials, an additional *stop* signal appears after some delay, which instructs the subject to withhold the discrimination or *go* response. As might be expected, the later the stop signal appears, the harder it is for subjects to stop the response [9] (see Figure 3). The classical model of the stop-signal task is the race model [11], which posits a race to threshold between *independent go* and *stop* processes. It also hypothesizes a stopping latency, the *stop-signal reaction time* (SSRT), which is the delay between stop signal onset and successful withholding of a *go* response. The (unobservable) SSRT is estimated as shown in Figure 1A, and is

thought to be longer in patient populations associated with inhibitory deficit than in healthy controls (attention-deficit hyperactivity disorder [1], obsessive-compulsive disorder [12], and substance dependence [13]). Some evidence suggests a neural correlate of the SSRT [8, 14, 5]. Although the race model is elegant in its simplicity and captures key experimental data, it is descriptive in nature and does not address how the stopping latency and other elements of the model depend on various underlying cognitive factors. Consequently, it cannot explain *why* behavior and stopping latency varies systematically across different experimental conditions or across different subject populations.

We present a normative, optimal decision-making framework for inhibitory control. We formalize interactions among various cognitive components: the continual monitoring of noisy sensory information, the integration of sensory inputs with top-down expectations, and the assessment of the relative values of potential actions. Our model has two principal components: (1) a *monitoring process*, based on Bayesian statistical inference, that infers the go stimulus identity within each trial, as well as task parameters across trials, (2) a *decision process*, formalized in terms of stochastic control, that translates current belief state based on sensory inputs into a moment-by-moment valuation of whether to choose one of the two go responses, or to wait longer. Given a certain belief state, the relative values of the various actions depend both on experimental parameters, such as the fraction of stop trials and the difficulty of go stimulus discrimination, as well as subject-specific parameters, such as learning rate and subjective valuation of rewards and costs. Within our normative model of inhibitory control, stopping latency is an *emergent property*, arising from interactions between the monitoring and decision processes. We show that our model captures classical behavioral data in the task, makes quantitative behavioral predictions under different experimental manipulations, and suggests that the brain may be implementing near-optimal decision-making in the stop-signal task.

## 2  Sensory processing as Bayes-optimal statistical inference

We model sensory processing in the stop-signal task as Bayesian statistical inference. In the *generative model* (see Figure 1B for graphical model), there are two independent hidden variables, corresponding to the identity of the go stimulus, $d \in \{0,1\}$, and whether or not the current trial is a stop trial, $s \in \{0,1\}$. Priors over $d$ and $s$ reflect experimental parameters: e.g. $P(d=1) = .5$, $P(s=1) = .25$ in typical stop signal experiments. Conditioned on $d$, a stream of iid inputs are generated on each trial, $x^1, \ldots, x^t, \ldots$, where $t$ indexes small increments of time within a trial, $p(x^t|d=0) = f_0(x^t)$, and $p(x^t|d=1) = f_1(x^t)$. For simplicity, we assume $f_0$ and $f_1$ to be Bernoulli distributions with distinct rate parameters $q_d$ and $1-q_d$, respectively. The dynamic variable $z^t$ denotes the presence/absence of the stop signal: if the stop signal appears at time $\theta$ then $z^1 = \ldots = z^{\theta-1} = 0$ and $z^\theta = z^{\theta+1} = \ldots = 1$. On a go trial, $s=0$, the stop-signal of course never appears, $P(\theta = \infty) = 1$. On a stop trial, $s=1$, we assume for simplicity that the onset of the stop signal follows a constant hazard rate, i.e. $\theta$ is generated from an exponential distribution: $p(\theta|s=1) = \lambda e^{-\lambda\theta}$. Conditioned on $z^t$, there is a separate iid stream of observations associated with the stop signal: $p(y^t|z^t=0) = g_0(y^t)$, and $p(y^t|z^t=1) = g_1(y^t)$. Again, we assume for simplicity that $g_0$ and $g_1$ are Bernoulli distributions with distinct rate parameters $q_s$ and $1-q_s$, respectively.

In the *recognition model*, the posterior probability associated with signal identity $p_d^t \triangleq P(d=1|\mathbf{x}^t)$, where $\mathbf{x}^t \triangleq \{x^1, \ldots, x^t\}$ denotes all the data observed so far, can be computed via Bayes' Rule:

$$p_d^t = \frac{p_d^{t-1} f_1(x^t)}{p_d^{t-1} f_1(x^t) + (1-p_d^{t-1})f_0(x^t)} = \frac{p_d^0 \Pi_{i=1}^t f_1(x^i)}{p_d^0 \Pi_{i=1}^t f_1(x^i) + (1-p_d^0)\Pi_{i=1}^t f_0(x^i)}$$

Inference about the stop signal is slightly more complicated due to the dynamics in $z^t$. First, we define $p_z^t$ as the posterior probability that the stop signal has already appeared $p_z^t \triangleq P\{\theta \leq t|\mathbf{y}^t\}$, where $\mathbf{y}^t \triangleq \{y^1, \ldots, y^t\}$. It can also be computed iteratively:

$$p_z^t = \frac{g_1(y^t)(p_z^{t-1} + (1-p_z^{t-1})h(t))}{g_1(y^t)(p_z^{t-1} + (1-p_z^{t-1})h(t)) + g_0(y^t)(1-p_z^{t-1})(1-h(t))}$$

where $h(t)$ is the posterior probability that the stop-signal will appear in the next instant given it has not appeared already, $h(t) \triangleq P(\theta=t|\theta > t-1, \mathbf{y}^{t-1})$.

$$h(t) = \frac{r \cdot P(\theta = t|s=1)}{r \cdot P(\theta > t-1|s=1) + (1-r)} = \frac{r\lambda e^{-\lambda t}}{re^{-\lambda(t-1)} + (1-r)}$$

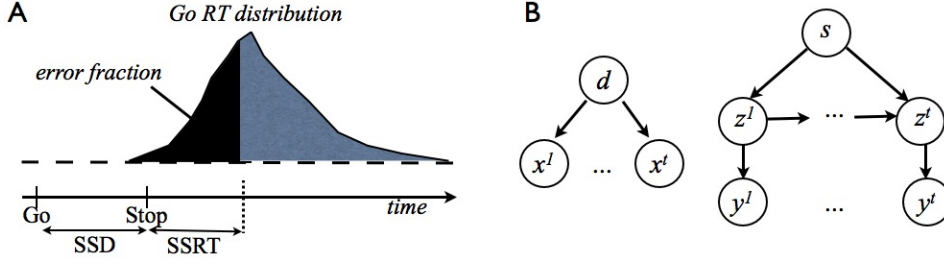

Figure 1: Modeling inhibitory control in the stop-signal task. (A) shows the race model for behavior in the stop-signal task [11]. Go and stop stimuli, separated by a stop signal delay (SSD), initiate two *independent* processes that race to thresholds and determine trial outcome. On go trials, noise in the go process results in a broad distribution over threshold-crossing times, i.e., the go reaction time (RT) distribution. The stop process is typically modeled as deterministic, with an associated *stop signal reaction time* or SSRT. The SSRT determines the fraction of go responses successfully stopped: the go RT cumulative density function evaluated at SSD+SSRT should give the stopping error rate at that SSD. Based on these assumptions, the SSRT is estimated from data given the go RT distribution, and error rate as a function of SSD. (B) Graphical model for sensory input generation in our Bayesian model. Two separate streams of observations, $\{x^1, \ldots, x^t, \ldots\}$ and $\{y^1, \ldots, y^t, \ldots\}$, are associated with the go and stop stimuli, respectively. $x^t$ depend on the identity of the target, $d \in \{0,1\}$. $y^t$ depends on whether the current trial is a stop trial, $s = \{0,1\}$, and whether the stop-signal has already appeared by time $t$, $z^t \in \{0,1\}$.

where $r = P(s=1)$ is the prior probability of a stop trial. Note that $h(t)$ does not depend on the observations, since *given* that the stop signal has not yet appeared, whether it will appear in the next instant does not depend on previous observations.

In the stop-signal task, a stop trial is considered a stop trial even if the subject makes the go response early, before the stop signal is presented. Following this convention, we need to compute the posterior probability that the current trial is a stop trial, denoted $p_s^t$, which depends both on the current belief about the presence of the stop signal, and the expectation that it will appear in the future:

$$p_s^t \triangleq P(s=1|\mathbf{y}^t) = p_z^t \cdot 1 + (1 - p_z^t) \cdot P(s=1|\theta > t, \mathbf{y}^t)$$

where $P(s=1|\theta>t, \mathbf{y}^t) = P(s=1|\theta>t)$ again does not depend on past observations:

$$P(s=1|\theta>t) = \frac{P(\theta>t|s=1)P(s=1)}{P(\theta>t|s=1)P(s=1) + P(\theta>t|s=0)P(s=0)} = \frac{e^{-\lambda t} \cdot r}{e^{-\lambda t} \cdot r + 1 \cdot (1-r)}$$

Finally, we define the belief state at time $t$ to be the vector $\mathbf{b}^t = (p_d^t, p_s^t)$.

Figure 2A shows the evolution of belief states for various trial types: (1) go trials, where no stop signal appears, (2) stop error (SE) trials, where a stop signal is presented but a response is made, and (3) stop success (SS) trials, where the stop signal is successfully processed to cancel the response. For simplicity, only trials where $d = 1$ are shown, and $\theta_s$ on stop trials is 17 steps. Due to stochasticity in the sensory information, the go stimulus is processed slower and the stop signal is detected faster than average on some trials; these lead to successful stopping, with SE trials showing the opposite trend. On all trials, $p_s$ shows an initial increase due to anticipation of the stop signal. Parameters used for the simulation were chosen to approximate typical experimental conditions (see e.g., Figure 3), and kept constant throughtout except where explicitly noted. The results do not change qualitatively when these settings are varied (data not shown).

## 3 Decision making as optimal stochastic control

In order to understand behavior as optimal decision-making, we need to specify a loss function that captures the reward structure of the task. We assume there is a deadline $D$ for responding on go trials, and an opportunity cost of $c$ per unit time on each trial. In addition, there is a penalty $c_s$ for choosing to respond on a stop-signal trial, and a unit cost for making an error on a go trial (by

choosing the wrong discrimination response or exceeding the deadline for responding). Because only the relative costs matter in the optimization, we can normalize the coefficients associated with all the costs such that one of them is unit cost. Let $\tau$ denote the trial termination time, so that $\tau = D$ if no response is made before the deadline, and $\tau < D$ if a response is made. On each trial, the policy $\pi$ produces a stopping time $\tau$ and a possible binary response $\delta \in \{0, 1\}$. The loss function is:

$$l(\tau, \delta; d, s, \theta, D) = c\tau + c_s \mathbf{1}_{\{\tau < D, s=1\}} + \mathbf{1}_{\{\tau < D, \delta \neq d, s=0\}} + \mathbf{1}_{\{\tau = D, s=0\}}$$

where $\mathbf{1}_{\{\cdot\}}$ is the indicator function. The optimal decision policy minimizes the average or *expected* loss, $L_\pi \triangleq \langle l(\tau, \delta; d, s, D) \rangle$,

$$L_\pi = c\langle \tau \rangle + c_s r P(\tau < D | s=1) + (1-r) P(\tau < D, \delta \neq d | s=0) + (1-r) P(\tau = D | s=0) \, .$$

Minimizing $L_\pi$ over the policy space directly is computationally intractable, but the dynamic programming principle provides an iterative relationship, the optimality equation, in terms of the *value function* (defined in terms of costs here), $V^t(\mathbf{b^t})$

$$V^t(\mathbf{b}^t) = \min_a \left[ \int p(\mathbf{b}^{t+1} | \mathbf{b}^t; a) V^{t+1}(\mathbf{b}^{t+1}) d\mathbf{b}^{t+1} \right] \, ,$$

where $a$ ranges over all possible actions. In our model, the action space consists of $\{go, wait\}$, with the corresponding expected costs (also known as Q-factors), $Q_g^t(\mathbf{b}^t)$ and $Q_w^t(\mathbf{b}^t)$, respectively.

$$
\begin{aligned}
Q_g^t(\mathbf{b}^t) &= ct + c_s p_s^t + (1 - p_s^t) min(p_d^t, 1 - p_d^t) \\
Q_w^t(\mathbf{b}^t) &= \mathbf{1}_{\{D > t+1\}} \langle V^{t+1}(\mathbf{b}^{t+1}) | \mathbf{b}^t \rangle_{\mathbf{b}^{t+1}} + \mathbf{1}_{\{D = t+1\}}(c(t+1) + 1 - p_s^t) \\
V^t(\mathbf{b}^t) &= \min(Q_g^t, Q_w^t)
\end{aligned}
$$

The value function is the smaller of the Q-factors $Q_g^t$ and $Q_w^t$, and the optimal decision policy chooses the action corresponding to the smallest Q-factor. Note that the *go* action results in either $\delta = 1$ or $\delta = 0$, depending on whether $p_d^\tau$ is greater or smaller than .5, respectively. The dependence of $Q_w^t$ on $V^{t+1}$ allows us to recursively compute the value function backwards in time. Given $V^{t+1}$, we can compute $\langle V^{t+1} \rangle$ by summing over the uncertainty about the next observations $x^{t+1}$, $y^{t+1}$, since the belief state $\mathbf{b}^{t+1}$ is a deterministic function of $\mathbf{b}^t$ and the observations.

$$
\begin{aligned}
\langle V^{t+1}(\mathbf{b}^{t+1}) | \mathbf{b}^t \rangle_{\mathbf{b}^t} &= \sum_{x^{t+1}, y^{t+1}} p(x^{t+1}, y^{t+1} | \mathbf{b}^t) V^{t+1}(\mathbf{b}^{t+1}(\mathbf{b}^t, x^{t+1}, y^{t+1})) \\
p(x^{t+1}, y^{t+1} | \mathbf{b}^t) &= p(x^{t+1} | p_d^t) p(y^{t+1} | p_s^t) \\
p(x^{t+1} | p_d^t) &= p_d^t f_1(x^{t+1}) + (1 - p_d^t) f_0(x^{t+1}) \\
p(y^{t+1} | p_s^t) &= (p_z^t + (1 - p_z^t) h(t+1)) g_1(y^{t+1}) + (1 - p_z^t)(1 - h(t+1)) g_0(y^{t+1})
\end{aligned}
$$

The initial condition of the value function can be computed exactly at the deadline since there is only one outcome (subject is no longer allowed to go or stop): $V^D(\mathbf{b}^D) = cD + (1 - p_s^D)$. We can then compute $\{V^t\}_{t=1}^D$ and the corresponding optimal decision policy backwards in time from $t = D - 1$ to $t = 1$. In our simulations, we do so numerically by discretizing the probability space for $p_s^t$ into 1000 bins; $p_d^t$ is represented exactly using its sufficient statistics. Note that dynamic programming is merely a convenient tool for computing the exact optimal policy. Our results show that humans and animals behave in a manner consistent with the optimal policy, indicating that the brain must use computations that are similar in nature. The important question of how such a policy may be computed or approximated neurally will be explored in future work.

Figure 2B demonstrates graphically how the Q-factors $Q_g$, $Q_w$ evolve over time for the trial types indicated in Figure 2A. Reflecting the sensory processing differences, SS trials show a slower drop in the cost of going, and a faster increase after the stop signal is processed; this is the converse of stop error trials. Note that although the average trajectory $Q_g$ does not dip below $Q_w$ in the non-canceled (error) stop trials, there is substantial variability in the individual trajectories under a Bernoulli observation model, and each one of them dips below $Q_w$ at some point. The histograms show reaction time distributions for go and SE trials.

## 4   Results

### 4.1   Model captures classical behavioral data in the stop-signal task

We first show that our model captures the basic behavioral results characteristic of the stop-signal task. Figure 3 compares our model predictions to data from Macaque monkeys performing a version

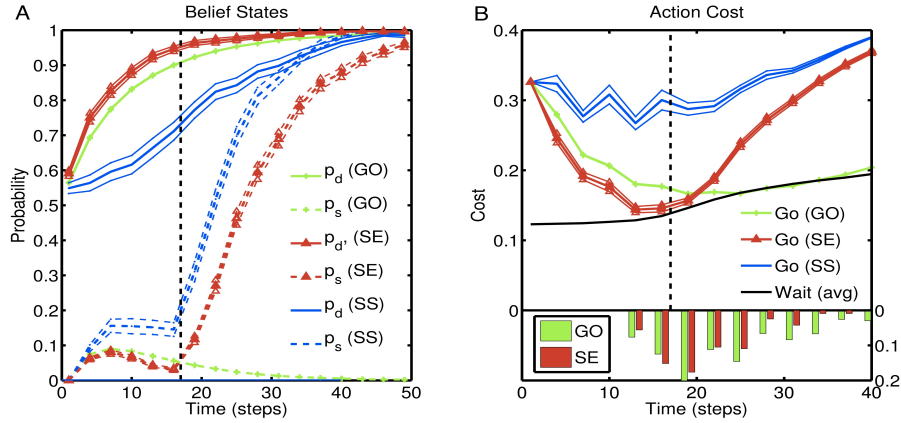

Figure 2: Mean trajectories of posteriors and Q-factors. (A) Evolution of the average belief states $p_d$ and $p_s$ corresponding to go and stop signals, for various trials–GO: go trials, SS: stop trials with successfully canceled response, SE: stop error trials. Stochasticity results in faster or slower processing of the two sensory input streams; these lead to stop success or error. For simplicity, $d = 1$ for all trials in the figure. The stop signal is presented at $\theta_s = 17$ time steps (dashed vertical line); the initial rise in $p_s$ corresponds to anticipation of a potential stop signal. (B) Go and Wait costs for the same partitioning of trials, along with the reaction time distributions for go and SE trials. On SE trials, the cost of going drops faster, and crosses below the cost of waiting before the stop signal can be adequately processed. Although the average go cost does not drop below the average wait cost, each individual trajectory crosses over at various time points, as indicated by the RT histograms. Simulation parameters: $q_d = 0.68, q_s = 0.72, \lambda = 0.1, r = 0.25, D = 50$ steps, $c_s = 50 * c$, where $c = 0.005$ per time step. $c$ is approximately the rate at which monkeys earn rewards in the task, which is equivalent to assuming that the cost of time (opportunity cost) should be set by the reward rate. Unless otherwise stated, these parameters are used in all the subsequent simulations. Thickness of lines indicates standard errors of the mean.

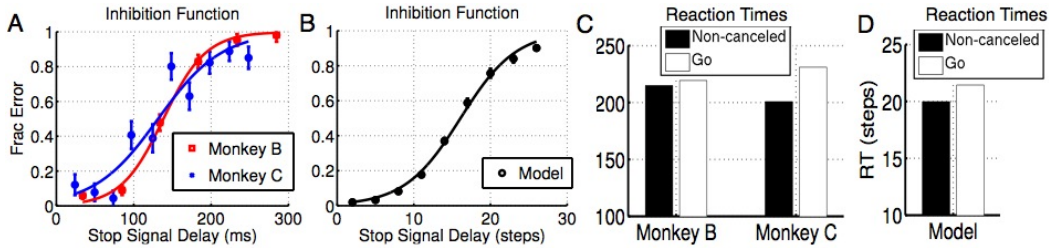

Figure 3: Optimal decision-making model captures classical behavioral effects in the stop-signal task. (A) Inhibition function: errors on stop trials increase as a function of SSD. (B) Effect reproduced by our model. (C) Discrimination RT is faster on non-canceled stop trials than go trials. (D) Effect reproduced by our model. (A,C) Data of two monkeys performing the stopping task (from [9]).

of the stop-signal task [9]. One of the basic measures of performance is the *inhibition function*, which is the average error rate on stop trials as a function of SSD. Error increases as SSD increases, as shown in the monkeys' behavior and also in our model (Figure 3A;B). Another classical result in the stop-signal task is that RT's on non-canceled (error) stop trials are on average faster than those on go trials (Figure 3C). Our model also reproduces this result (Figure 3D). Intuitively, this is because inference about the go stimulus identity can proceed slowly or rapidly on different trials, due to noise in the observation process. Non-canceled trials are those in which $p_d$ happens to evolve rapidly enough for a go response to be initiated before the stop signal is adequately processed. Go trial RT's, on the other hand, include all trajectories, whether $p_d$ happens to evolve quickly or not (see Figure 2).

## 4.2 Effect of stop trial frequency on behavior

The overall frequency of stop signal trials has systematic effects on stopping behavior [6]. As the fraction of stop trials is increased, go responses slow down and stop errors decrease in a graded fashion (Figure 4A;B). In our model (Figure 4C;D), the stop signal frequency, $r$, influences the speed with which a stop signal is detected, whereby larger $r$ leads to greater posterior belief that a stop signal is present, and also greater confidence that a stop signal will appear soon even it has not already. It therefore controls the tradeoff between going and stopping in the optimal policy. If stop signals are more prevalent, the optimal decision policy can use that information to make fewer errors on stop trials, by delaying the go response, and by detecting the stop signal faster.

Even in experiments where the fraction of stop trials is held constant, chance runs of stop or go trials may result in fluctuating local frequency of stop trials, which in turn may lead to trial-by-trial behavioral adjustments due to subjects' fluctuating estimate of $r$. Indeed, subjects speed up after a chance run of go trials, and slow down following a sequence of stop trials [6] (see Figure 4E). We model these effects by assuming that subjects believe that the stop signal frequency $r_k$ on trial $k$ has probability $\alpha$ of being the same as $r_{k-1}$ and probability $1 - \alpha$ of being re-sampled from a prior distribution $p_0(r)$, chosen in our simulations to be a beta distribution with a bias toward small $r$ (infrequent stop trials). Previous work has shown that this is essentially equivalent to using a causal, exponential window to estimate the current rate of stop trials [20], where the exponential decay constant is monotonically related to the assumed volatility in the environment in the Bayesian model. The probability of trial $k$ being a stop trial, $P(s_k = 1 | \mathbf{s}_{k-1})$, where $\mathbf{s}_k \triangleq \{s_1, \ldots, s_k\}$, is

$$P(s_k = 1 | \mathbf{s}_{k-1}) = \int P(s_k = 1 | r_k) p(r_k | \mathbf{s}_{k-1}) dr_k = \int r_k p(r_k | \mathbf{s}_{k-1}) dr_k = \langle r_k | \mathbf{s}_{k-1} \rangle .$$

In other words, the predictive probability of seeing a stop trial is just the mean of the predictive distribution $p(r_k | \mathbf{s}_{k-1})$. We denote this mean as $\hat{r}_k$. The predictive distribution is a mixture of the previous posterior distribution and a fixed prior distribution, with $\alpha$ and $1 - \alpha$ acting as the mixing coefficients, respectively:

$$p(r_k | \mathbf{s}_{k-1}) = \alpha p(r_{k-1} | \mathbf{s}_{k-1}) + (1 - \alpha) p_0(r_k)$$

and the posterior distribution is updated according to Bayes' Rule:

$$p(r_k | \mathbf{s}_k) \propto P(s_k | r_k) p(r_k | \mathbf{s}_{k-1}) .$$

As shown in Figure 4F, our model successfully explains observed sequential effects in behavioral data. Since the majority of trials (75%) are go trials, a chance run of go trials impacts RT much less than a chance run of stop trials. The figure also shows results for different values of $\alpha$, with all other parameters kept constant. These values encode different expectations about *volatility* in the stop trial frequency, and produce slightly different predictions about sequential effects. Thus, $\alpha$ may be an important source of individual variability observed in the data, along with the other model parameters.

Recent data shows that neural activity in the supplementary eye field is predictive of trial-by-trial slowing as a function of the recent stop trial frequency [15]. Moreover, microstimulation of supplementary eye field neurons results in slower responses to the go stimulus and fewer stop errors [16]. Together, this suggests that supplementary eye field may encode the local frequency of stop trials, and influence stopping behavior in a statistically appropriate manner.

## 4.3 Influence of reward structure on behavior

The previous section demonstrated how adjustments to behavior in the face of experimental manipulations can be seen as instances of optimal decision-making in the stop signal task. An important component of the race model for stopping behavior [11] is the SSRT, which is thought to be a stable, subject-specific index of stopping ability. In this section, we demonstrate that the SSRT can be seen as an *emergent property* of optimal decision-making, and is consequently modified in predictable ways by experimental manipulation.

Leotti & Wager showed that subjects can be biased toward stopping or going when the relative penalties associated with go and stop errors are experimentally manipulated [10]. Figure 5A;B show that as subjects are biased toward stopping, they make fewer stop trial errors and have slower

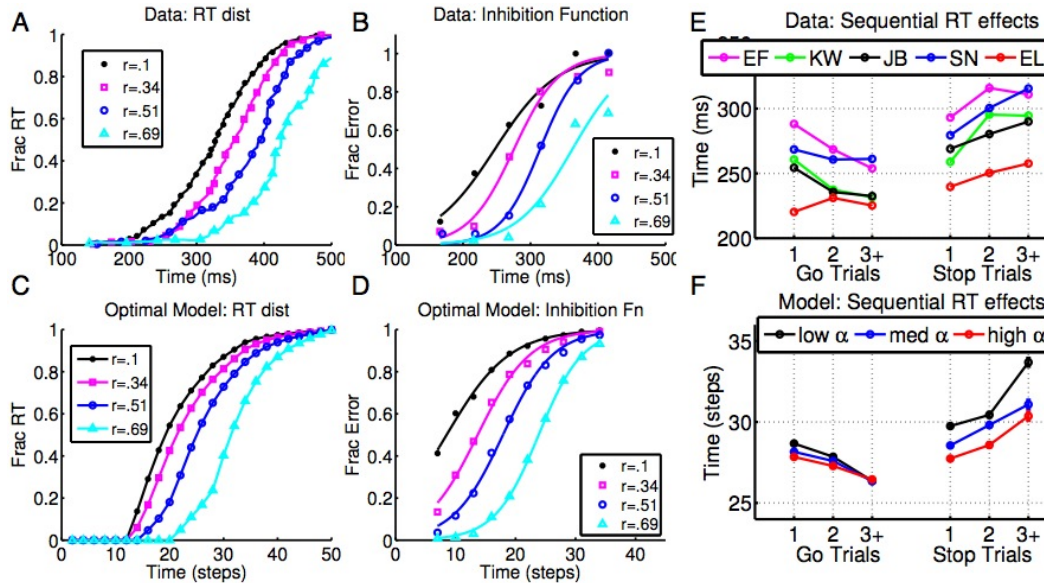

Figure 4: Effect of global and local frequency of stop trials on behavior. We compare model predictions with experimental data from a monkey performing the stop-signal task (adapted from Emeric et al., 2007). (A) Go reaction times shift to the right (slower), as the fraction of stop trials is increased. (B) Inhibitory function (stop error rate as a function of SSD) shifts to the right (fewer errors), as the fraction of stop trials is increased. Data adapted from [6]. (C;D) Our model predicts similar effects. (E) Sequential effects in reaction times from 6 subjects showing faster go RTs following longer sequences of go trials (columns 1-3), and slower RTs following longer sequences of stop trials (columns 4-6, data adapted from [6]). (F) Our model reproduces these changes; the parameter $\alpha$ controls the responsiveness to trial history, and may explain inter-subject differences. Values of alpha: low=0.85, med=0.95, high=0.98.

go responses. Our model reproduces this behavior when $c_s$, the parameter representing the cost of stopping, is set to small, medium and high values. Increasing the cost of a stop error induces an increase in reaction time and an associated decrease in the fraction of stop errors. This is a direct consequence of the optimal model attempting to minimize the total expected cost – with stop errors being more expensive, there is an incentive to slow down the go response in order to minimize the possibility of missing a stop signal.

Critically, the SSRT in the human data *decreases* with increasing bias toward stopping (Figure 5C). Although the SSRT is not an explicit component of our model, we can nevertheless estimate it from the reaction times and fraction of stop errors produced by our model simulations, following the race model's prescribed procedure [11]. Essentially, the SSRT is estimated as the difference between mean go RT and the SSD at which 50% stop errors are committed (see Figure 1). By reconciling the competing demands of stopping and going in an optimal manner, the estimated SSRT from our simulations is automatically adjusted to mimic the observed human behavior (Figure 5F). This suggests that the SSRT emerges naturally out of rational decision-making in the task.

## 5 Discussion

We presented an optimal decision-making model for inhibitory control in the stop-signal task. The parameters of the model are either set directly by experimental design (cost function, stop frequency and timing), or correspond to subject-specific abilities that can be estimated from behavior (sensory processing); thus, there are no "free" parameters. The model successfully captures classical behavioral results, such as the increase in error rate on stop trials with the increase of SSD, as well as the decreases in average response time from go trials to error stop trials. The model also captures more subtle changes in stopping behavior, when the fraction of stop-signal trials, the penalties for various types of errors, and the history of experienced trials are manipulated. The classical model for the task

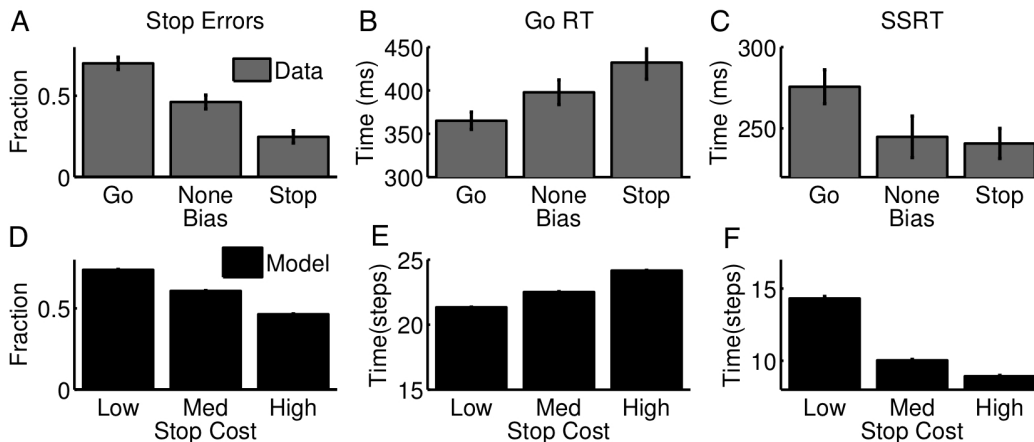

Figure 5: Effect of reward on stopping. (A-C) Data from human subjects performing a variant of the stop-signal task where the ratio of rewards for quick go responses and successful stopping was varied, inducing a bias towards going or stopping (Data from [10]). An increased bias towards stopping (i.e., fewer stop errors, (A)) is associated with an increase in the average reaction time on go trials (B), and a *decrease* in the stopping latency or SSRT (C). (D-F) Our model captures this change in SSRT as a function of the inherent tradeoff between RT and stop errors. Values of $c_s$: low=0.15, med=0.25, high=0.5.

(the race model) does not directly explain or quantitatively predict these changes in behavior. Moreover, the stopping latency measure prescribed by the race model (the SSRT) changes systematically across various experimental manipulations, indicating that it cannot be used as a simplistic, global measure of inhibitory control for each subject. Instead, inhibitory control is a multifaceted function of factors such as subject-specific sensory processing rates, attentional factors, and internal/external bias towards stopping or going, which are explicitly related to parameters in our normative model.

The close correspondence of model predictions with human and animal behavior suggests that the computations necessary for optimal behavior are exactly or approximately implemented by the brain. We used dynamic programming as a convenient tool to compute the optimal monitoring and decisional computations, but the brain is unlikely to use this computationally expensive method. Recent studies of the frontal eye fields (FEF, [8]) and superior colliculus [14] of monkeys show neural responses that diverge on go and correct stop trials, indicating that they may encode computations leading to the execution or cancellation of movement. It is possible that optimal behavior can be *approximated* by a diffusion process implementing the race model [4, 19], with the rate and threshold parameters adjusted according to task demands. In future work, we will study more explicitly how optimal decision-making can be approximated by a diffusion model implementation of the race model (see e.g., [18], and how the parameters of such an implementation may be set to reflect task demands. We will also assess alternatives to the race model, in the form of other approximate algorithms, in terms of their ability to capture behavioral data and explain neural data.

One major aim of our work is to understand how stopping ability and SSRT arise from various cognitive factors, such as sensitivity to rewards, learning capacity related to estimating stop signal frequency, and the rate at which sensory inputs are processed. This composite view of stopping ability and SSRT may help explain group differences in stopping behavior, in particular, differences in SSRT observed in a number of psychiatric and neurological conditions, such as substance abuse [13], attention-deficit hyperactivity disorder [1], schizophrenia [3], obsessive-compulsive disorder [12], Parkinson's disease [7], Alzheimer's disease [2], *et cetera*. One of our goals for future research is to map group differences in stopping behavior to the parameters of our model, thus gaining insight into exactly which cognitive components go awry in each dysfunctional state.

# References

[1] R.M. Alderson, M.D. Rapport, and M.J. Kofler. Attention-deficit/hyperactivity disorder and behavioral inhibition: a meta-analytic review of the stop-signal paradigm. *Journal of Abnormal Child Psychology*, 35(5):745–758, 2007.

[2] H Amieva, S Lafont, S Auriacombe, N Le Carret, J F Dartigues, J M Orgogozo, and C Fabrigoule. Inhibitory breakdown and dementia of the Alzheimer type: A general phenomenon? *Journal of Clinical and Experimental Neuropsychology*, 24(4):503–516, 2992.

[3] J C Badcock, P T Michie, L Johnson, and J Combrinck. Acts of control in schizophrenia: Dissociating the components of inhibition. *Psychological Medicine*, 32(2):287–297, 2002.

[4] L. Boucher, T.J. Palmeri, G.D. Logan, and J.D. Schall. Inhibitory control in mind and brain: an interactive race model of countermanding saccades. *Psychological Review*, 114(2):376–397, 2007.

[5] CD Chambers, H Garavan, and MA Bellgrove. Insights into the neural basis of response inhibition from cognitive and clinical neuroscience. *Neuroscience and Biobehavioral Reviews*, 33(5):631–646, 2009.

[6] E.E. Emeric, J.W. Brown, L. Boucher, R.H.S. Carpenter, D.P. Hanes, R. Harris, G.D. Logan, R.N. Mashru, M. Paré, P. Pouget, V. Stuphorn, T.L. Taylor, and J Schall. Influence of history on saccade countermanding performance in humans and macaque monkeys. *Vision research*, 47(1):35–49, 2007.

[7] S Gauggel, M Rieger, and T Feghoff. Inhibition of ongoing responses in patients with Pakingson's disease. *J. Neurol. Neurosurg. Psychiatry*, (75):4, 539-544 2004.

[8] D.P. Hanes, W.F. Patterson, and J.D. Schall. The role of frontal eye field in countermanding saccades: Visual, movement and fixation activity. *Journal of Neurophysiology*, 79:817–834, 1998.

[9] DP Hanes and JD Schall. Countermanding saccades in macaque. *Visual Neuroscience*, 12(5):929, 1995.

[10] L.A. Leotti and T.D. Wager. Motivational influences on response inhibition measures. *J Exp Psychol Hum Percept Perform*, 2009.

[11] G.D. Logan and W.B. Cowan. On the ability to inhibit thought and action: A theory of an act of control. *Psychological Review*, 91(3):295–327, 1984.

[12] L. Menzies, S. Achard, S.R. Chamberlain, N. Fineberg, C.H. Chen, N. del Campo, B.J. Sahakian, T.W. Robbins, and E. Bullmore. Neurocognitive endophenotypes of obsessive-compulsive disorder. *Brain*, 130(12):3223, 2007.

[13] J.T. Nigg, M.M. Wong, M.M. Martel, J.M. Jester, L.I. Puttler, J.M. Glass, K.M. Adams, H.E. Fitzgerald, and R.A. Zucker. Poor response inhibition as a predictor of problem drinking and illicit drug use in adolescents at risk for alcoholism and other substance use disorders. *Journal of Amer Academy of Child & Adolescent Psychiatry*, 45(4):468, 2006.

[14] M. Pare and D.P. Hanes. Controlled movement processing: superior colliculus activity associated with countermanded saccades. *Journal of Neuroscience*, 23(16):6480–6489, 2003.

[15] V. Stuphorn, J.W. Brown, and J.D. Schall. Role of Supplementary Eye Field in Saccade Initiation: Executive, Not Direct, Control. *Journal of Neurophysiology*, 103(2):801, 2010.

[16] V. Stuphorn and J.D. Schall. Executive control of countermanding saccades by the supplementary eye field. *Nature neuroscience*, 9(7):925–931, 2006.

[17] F. Verbruggen and G.D. Logan. Models of response inhibition in the stop-signal and stop-change paradigms. *Neuroscience & Biobehavioral Reviews*, 33(5):647–661, 2009.

[18] F. Verbruggen and G.D. Logan. Proactive adjustments of response strategies in the stop-signal paradigm. *Journal of Experimental Psychology: Human Perception and Performance*, 35(3):835–854, 2009.

[19] K.F. Wong-Lin, P. Eckhoff, P. Holmes, and J.D. Cohen. Optimal performance in a countermanding saccade task. *Brain Research*, 2009.

[20] AJ Yu and JD Cohen. Sequential effects: Superstition or rational behavior? *Advances in Neural Information Processing Systems*, 21:1873–1880, 2009.

